# Amplifying and Linearizing Apical Synaptic Inputs to Cortical Pyramidal Cells.

**Öjvind Bernander, Christof Koch** *
Computation and Neural Systems Program,
California Institute of Technology, 139-74
Pasadena, Ca 91125, USA.

**Rodney J. Douglas**
Anatomical Neuropharmacology Unit,
Dept. Pharmacology,
Oxford, UK.

## Abstract

Intradendritic electrophysiological recordings reveal a bewildering repertoire of complex electrical spikes and plateaus that are difficult to reconcile with conventional notions of neuronal function. In this paper we argue that such dendritic events are just an exuberant expression of a more important mechanism – a proportional *current* amplifier whose primary task is to offset electrotonic losses. Using the example of functionally important synaptic inputs to the superficial layers of an anatomically and electrophysiologically reconstructed layer 5 pyramidal neuron, we derive and simulate the properties of conductances that linearize and amplify distal synaptic input current in a graded manner. The amplification depends on a potassium conductance in the apical tuft and calcium conductances in the apical trunk.

*To whom all correspondence should be addressed.

# 1   INTRODUCTION

About half the pyramidal neurons in layer 5 of neocortex have long apical dendrites that arborize extensively in layers 1-3. There the dendrites receive synaptic input from the inter-areal feedback projections (Felleman and van Essen, 1991) that play an important role in many models of brain function (Rockland and Virga, 1989). At first sight this seems to be an unsatisfactory arrangement. In light of traditional passive models of dendritic function the distant inputs cannot have a significant effect on the output discharge of the pyramidal cell. The distal inputs are at least one to two space constants removed from the soma in layer 5 and so only a small fraction of the voltage signal will reach there. Nevertheless, experiments in cortical slices have shown that synapses located in even the most superficial cortical layers can provide excitation strong enough to elicit action potentials in the somata of layer 5 pyramidal cells (Cauller and Connors, 1992, 1994). These results suggest that the apical dendrites are active rather than passive, and able to amplify the signal *en route* to the soma. Indeed, electrophysiological recordings from cortical pyramidal cells provide ample evidence for a variety of voltage-dependent dendritic conductances that could perform such amplification (Spencer and Kandel, 1961; Regehr et al., 1993; Yuste and Tank, 1993; Pockberger, 1991; Amitai et al., 1993; Kim and Connors, 1993).

Although the available experimental data on the various active conductances provide direct support for amplification, they are not adequate to specify the mechanism by which it occurs. Consequently, notions of dendritic amplification have been informal, usually favoring voltage gain, and mechanisms that have a binary (high gain) quality. In this paper, we formalize what conductance properties are required for a current amplifier, and derive the required form of their voltage dependency by analysis.

We propose that current amplification depends on two active conductances: a voltage-dependent $K^+$ conductance, $g_K$, in the superficial part of the dendritic tree that linearizes synaptic input, and a voltage-dependent $Ca^{2+}$ conductance, $g_{Ca}$, in layer 4 that amplifies the result of the linearization stage. Spencer and Kandel (1961) hypothesized the presence of dendritic calcium channels that amplify distal inputs. More recently, a modeling study of a cerebellar Purkinje cell suggests that dendritic calcium counteracts attenuation of distal inputs so that the somatic response is independent of synaptic location (De Schutter and Bower, 1992). A gain-control mechanism involving both potassium and calcium has also been proposed in locust non-spiking interneurons (Laurent, 1993). In these cells, the two conductances counteract the nonlinearity of graded transmitter release, so that the output of the interneuron was independent of its membrane voltage. The principle that we used can be explained with the help of a highly simplified three compartment model (Fig. 1A). The leftmost node represents the soma and is clamped to -50 mV. The justification for this is that the time-averaged somatic voltage is remarkably constant and close to -50 *mV* for a wide range of spike rates. The middle node represents the apical trunk containing $g_{Ca}$, and the rightmost node represents the apical tuft with a synaptic induced conductance change $g_{syn}$ in parallel with $g_K$. For simplicity we assume that the model is in steady-state, and has an infinite membrane resistance, $R_m$.

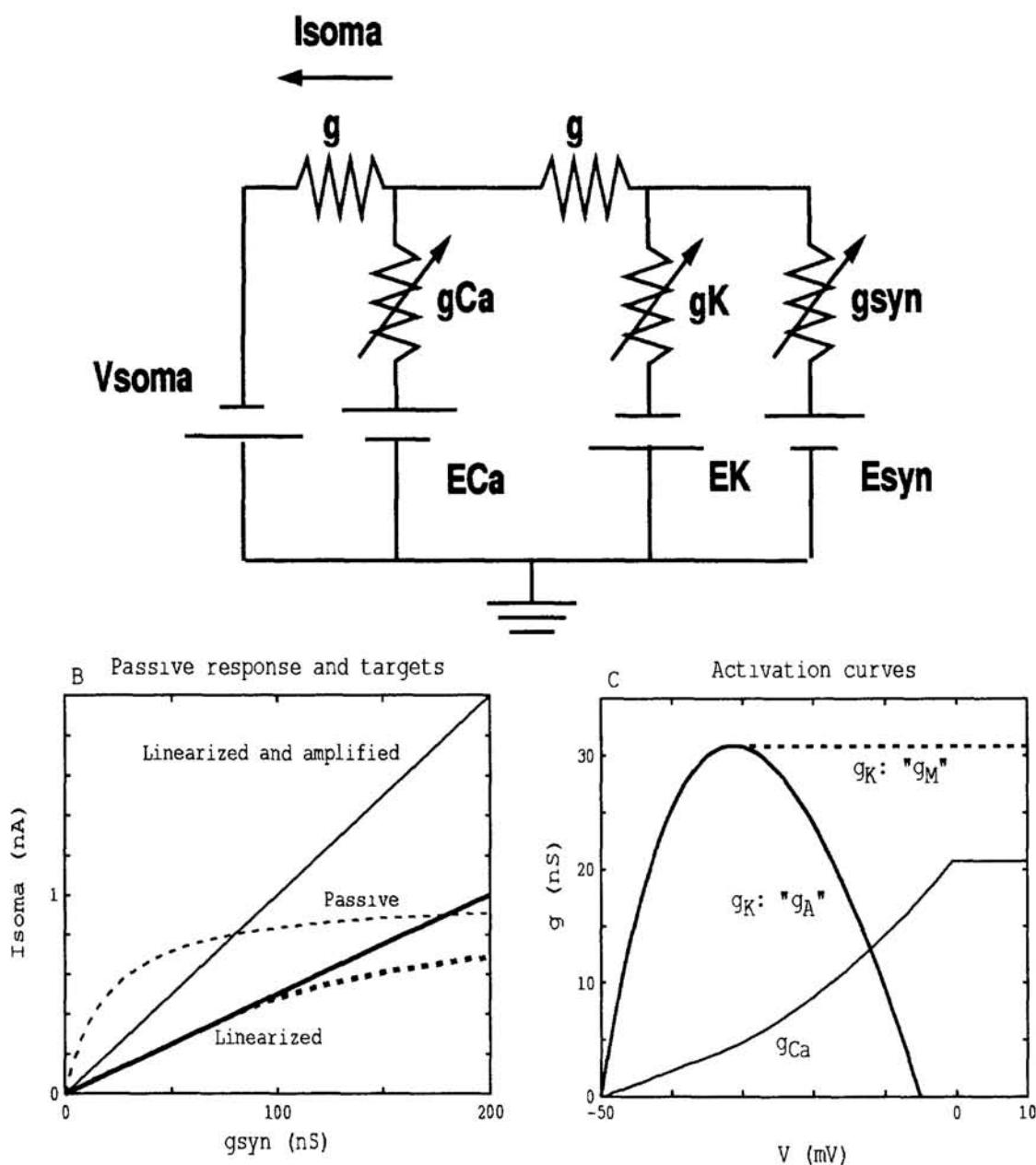

Figure 1: **Simplified model used to demonstrate the concepts of saturation, linearization, and amplification.** (**A**) Circuit diagram. The somatic compartment was clamped to $V_{soma} = -50\ mV$ with $E_{Ca} = 115\ mV$, $E_K = -95\ mV$, $E_{syn} = 0\ mV$, and $g = 40\ nS$. The membrane capacitance was ignored, since only steady state properties were studied, and membrane leak was not included for simplicity. (**B**) Somatic current, $I_{soma}$, in response to synaptic input. The passive response (thin dashed line) is sublinear and saturates for low values of $g_{syn}$. The linearized response (thick solid line) is obtained by introducing an inactivating potassium conductance, $G_K$ ("$g_A$" in c). A persistent persistent $G_K$ results in a somewhat sub-linear response (thick dashed line; "$g_M$" in c). The addition of a calcium conductance amplifies the response (thin solid line). (**C**) Analytically derived activation curves. The inactivating potassium conductance ("$I_A$") was derived, but the persistent version ("$I_M$") proved to be more stable.

## 2  RESULTS

Fig. 1B shows the computed relationship between the excitatory synaptic input conductance and the axial current, $I_{soma}$, flowing into the somatic (leftmost) compartment. The synaptic input rapidly saturates; increasing $g_{syn}$ beyond about 50 nS leads to little further increase in $I_{soma}$. This saturation is due to the EPSP in the distal compartment reducing the effective synaptic driving potential. We propose that the first goal of dendritic amplification is to linearize this relationship, so that the soma is more sensitive to the exact amount of excitatory input impinging on the apical tuft, by introducing a potassium conductance that provides a hyperpolarizing current in proportion to the degree of membrane depolarization. The voltage-dependence of such a conductance can be derived by postulating a linear relationship between the synaptic current flowing into the somatic node and the synaptic input, i.e. $I_{soma} = constant \cdot g_{syn}$. In conjunction with Ohm's law and current conservation, this relation leads to a simple fractional polynominal for the voltage dependency of $g_K$ (labeled "$g_A$" in Fig. 1C). As the membrane potential depolarizes, $g_K$ activates and pulls it back towards $E_K$. At large depolarizations $g_K$ inactivates, similar to the "A" potassium conductance, resulting overall in a linear relationship between input and output (Fig. 1B). As the slope conductance of this particular form of $g_K$ can become negative, causing amplification of the synaptic input, we use a variant of $g_K$ that is monotonized by leveling out the activation curve after it has reached its maximum, similar to the "M" current (Fig. 1C). Incorporating this non-inactivating $K^+$ conductance into the distal compartment results in a slightly sublinear relationship between input and output (Fig. 1B).

With $g_K$ in place, amplification of $I_{soma}$ is achieved by introducing an inward current between the soma and the postsynaptic site. The voltage-dependency of the amplification conductance can be derived by postulating $I_{soma} = gain \cdot constant \cdot g_{syn}$. This leads to the non-inactivating $g_{Ca}$ shown in Fig. 1C, in which the overall relationship between synaptic input and somatic output current (Fig. 1B) reflects the amplification.

We extend this concept of deriving the form of the required conductances to a detailed model of a morphologically reconstructed layer 5 pyramidal cell from cat visual cortex (Douglas *et al.*, 1991, Fig. 2A;). We assume a passive dendritic tree, and include a complement of eight common voltage-dependent conductances in its soma. 500 non-NMDA synapses are distributed on the dendritic tuft throughout layers 1, 2 and 3, and we assume a proportionality between the presynaptic firing frequency $f_{in}$ and the time-averaged synaptic induced conductance change. When $f_{in}$ is increased, the detailed model exhibits the same saturation as seen in the simple model (Fig. 2B). Even if all 500 synapses are activated at $f_{in} = 500\,Hz$ only 0.65 nA of current is delivered to the soma. This saturation is caused when the synaptic input current flows into the high input resistances of the distal dendrites, thereby reducing the synaptic driving potential. Layer 1 and 2 input together can contribute a maximum of 0.25 nA to the soma. This is too little current to cause the cell to spike, in contrast with the experimental evidence (Cauller and Connors, 1994), in which spike discharge was evoked reliably. Electrotonic losses make only a minor contribution to the small somatic signal. Even when the membrane leak current is eliminated by setting $R_m$ to infinity, $I_{soma}$ only increases a mere 2% to 0.66 nA.

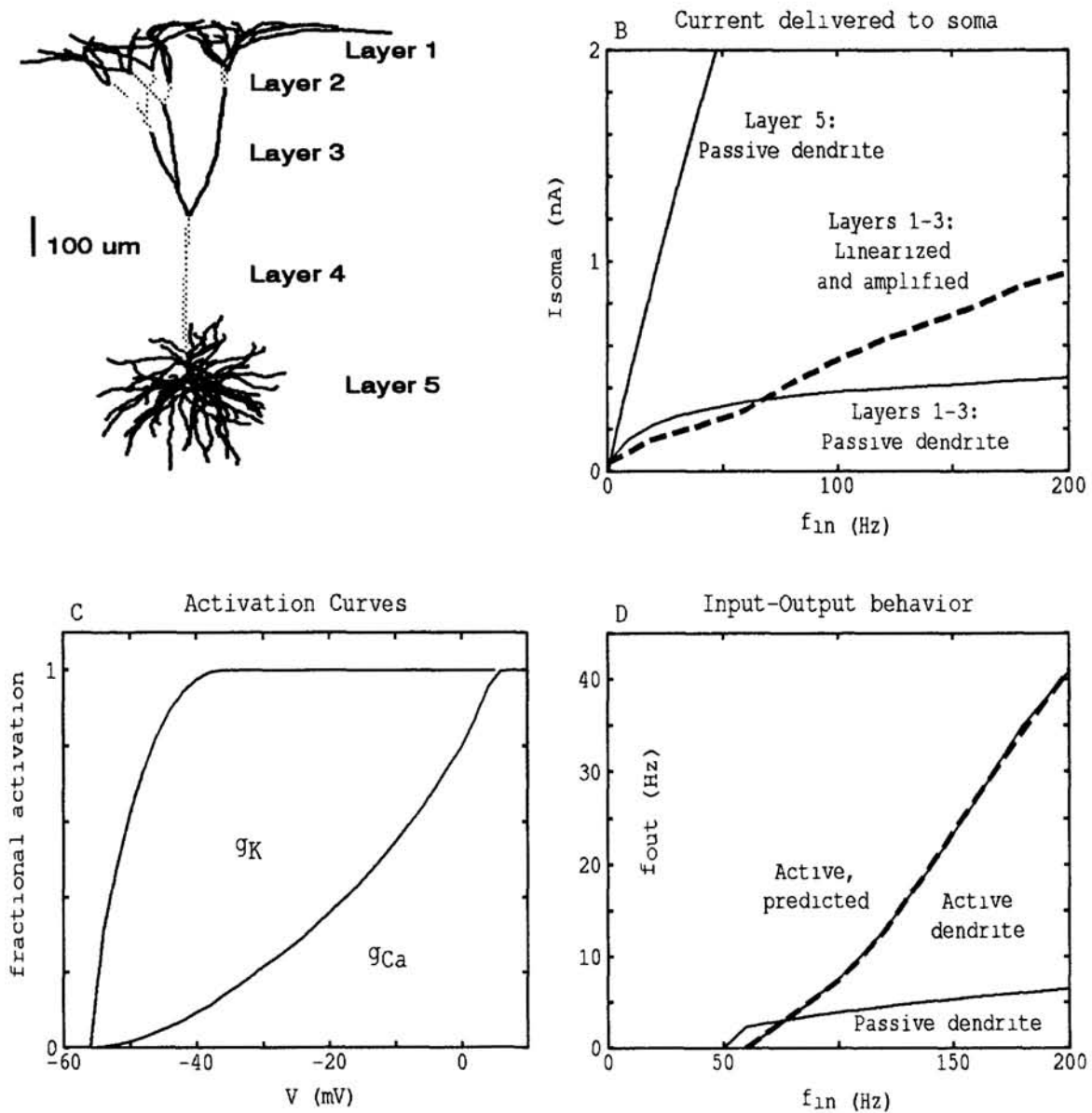

Figure 2: **Amplification in the detailed model. (A)** The morphology of this layer V pyramidal cell was reconstructed from a HRP-stained cell in area 17 of the adult cat (Douglas *et al.*, 1991). The layers are marked in alternating black and grey. The boundaries between superficial layers are not exact, but rough estimates and were chosen at branch points; a few basal dendrites may reach into layer 6. Axon not shown. **(B)** Current delivered to the soma by stimulation of 500 AMPA synapses throughout either layer 5 or layers 1–3. **(C)** Derived activation curves for $g_K$ and $g_{Ca}$. Sigmoidal fits of the form $g(V) = 1/(1 + e^{(V_{half} - V)/K})$, resulted in $K_K = 3.9\ mV$, $V_{half,K} = -51\ mV$, $K_{Ca} = 13.7\ mV$, $V_{half,Ca} = -14\ mV$. **(D)** Output spike rate as a function of input activation rate of 500 AMPA synapses in layers 1–3, with and without the derived conductances. The dashed line shows the $f_{out}$ rate predicted by using the linear target $I_{soma}$ as a function of $f_{in}$ in combination with the somatic $f - I$ relationship.

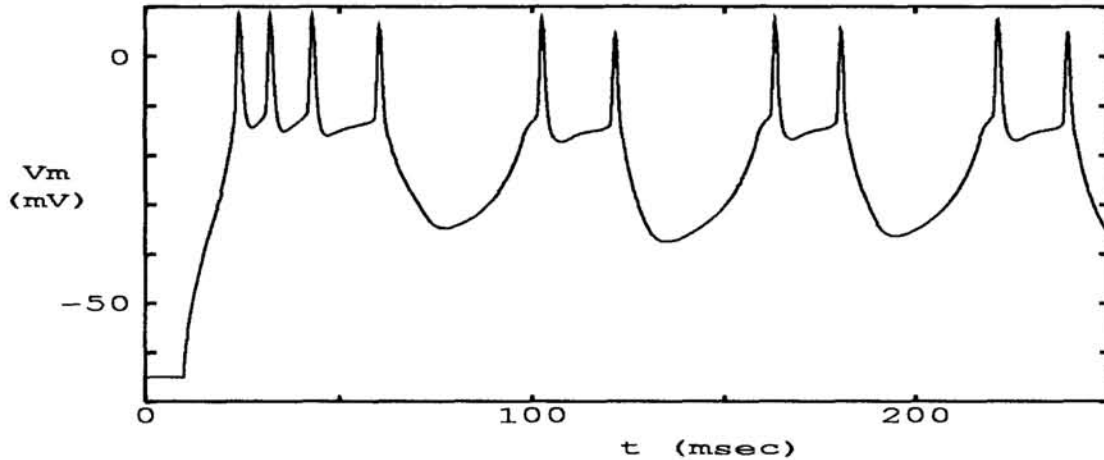

Figure 3: **Dendritic calcium spikes.** All-or-nothing dendritic $Ca^{2+}$ calcium spikes can be generated by adding a voltage-independent but $Ca^{2+}$-dependent $K^+$ conductance to the apical tree with $g_{max} = 11.4\,nS$. The trace shown is in response to sustained intradendritic current injection of 0.5 nA. For clamp currents of 0.3 nA or less, no calcium spikes are triggered and only single somatic spikes are obtained (not shown). These currents do not substantially affect the current amplifier effect.

By analogy with the simple model of Fig. 1, we eliminate the saturating response by introducing a non-inactivating form of $g_K$ spread evenly throughout layers 1-3. The resulting linearized response is amplified by a $Ca^{2+}$ conductance located at the base of the apical tuft, where the apical dendrite crosses from layer 4 to layer 3 (Fig. 2A). This is in agreement with recent calcium imaging experiments, which established that layer 5 neocortical pyramidal cells have a calcium hot spot in the apical tree about 500-600 $\mu m$ away from the soma (Tank *et al.*, 1988). Although the derivation of the voltage-dependency of these two conductances is more complicated than in the three compartment model, the principle of the derivation is similar (Bernander, 1993, Fig. 2C;). We derive a $Ca^{2+}$ conductance, for a synaptic current gain of two, resembling a non-inactivating, high-threshold calcium conductance. The curve relating synaptic input frequency and axial current flowing into the soma (Fig. 2B) shows both the linearized and amplified relationships.

Once above threshold, the model cell has a linear current-discharge relation with a slope of about 50 spikes per second per nA, in good agreement with experimental observations *in vitro* (Mason and Larkman, 1990) and *in vivo* (Ahmed *et al.*, 1993). Given a sustained synaptic input frequency, the somatic f-I relationship can be used to convert the synaptic current flowing into the soma $I_{soma}$ into an equivalent output frequency (Abbott, 1991; Powers *et al.*, 1992; Fig. 2D). This simple transformation accounts for all the relevant nonlinearities, including synaptic saturation, interaction and the threshold mechanism at the soma or elsewhere. We confirmed the validity of our transformation method by explicitly computing the expected relationship between $f_{in}$ and $f_{out}$, without constraining the somatic potential, and comparing the two. Qualitatively, both methods lead to very similar results (Fig. 2D): in the

presence of dendritic $g_{Ca}$ superficial synaptic input can robustly drive the cell, in a proportional manner over a large input range.

The amplification mechanism derived above is continuous in the input rate. It does not exhibit the slow calcium spikes described in the literature (Pockberger, 1991; Amitai *et al.*, 1993; Kim and Connors, 1993). However, it is straightforward to add a calcium-dependent potassium conductance yielding such spikes. Incorporating such a conductance into the apical trunk leads to calcium spikes (Fig. 3) in response to an intradendritic current injection of 0.4 nA or more, while for weaker inputs no such events are seen. In response to synaptic input to the tuft of 120 *Hz* or more, these spikes are activated, resulting in a moderate depression (25% or less) of the average output rate, $f_{out}$ (not shown).

In our view, the function of the dendritic conductances underlying this all-or-none *voltage* event is the gradual *current* amplification of superficial input, without amplifying synaptic input to the basal dendrites (Bernander, 1993). Because $g_{Ca}$ depolarizes the membrane, further activating $g_{Ca}$, the gain of the current amplifier is very sensitive to the density and shape of the dendritic $g_{Ca}$. Thus, neuromodulators that act upon $g_{Ca}$ control the extent to which cortical feedback pathways, acting via superficial synaptic input, have access to the output of the cell.

## Acknowledgements

This work was supported by the Office of Naval Research, the National Institute of Mental Health through the Center for Neuroscience, the Medical Research Council of the United Kingdom, and the International Human Frontier Science Program.

# References

[1] L.F. Abbott. Realistic synaptic inputs for model neuronal networks. *Network*, 2:245–258, 1991.

[2] B. Ahmed, J.C. Anderson, R.J. Douglas, K.A.C. Martin, and J.C. Nelson. The polyneuronal innervation of spiny steallate neurons in cat visual cortex. *Submitted*, 1993.

[3] Y. Amitai, A. Friedman, B.W. Connors, and M.J. Gutnick. Regenerative activity in apical dendrites of pyramidal cells in neocortex. *Cerebral Cortex*, 3:26–38, 1993.

[4] Ö Bernander. Synaptic integration and its control in neocortical pyramidal cells. May 1993. Ph.D. thesis, California Institute of Technology.

[5] L.J. Cauller and B.W. Connors. Functions of very distal dendrites: experimental and computational studies of layer I synapses on neocortical pyramidal cells. In T. McKenna, J. Javis, and S.F. Zarnetzer, editors, *Single Neuron Computation*, chapter 8, pages 199–229. Academic Press, Boston, MA, 1992.

[6] L.J. Cauller and B.W. Connors. *J. Neuroscience*, In Press.

[7] E. De Schutter and J.M. Bower. Firing rate of purkinje cells does not depend on the dendritic location of parallel fiber inputs. *Eur. J. of Neurosci.*, S5:17, 1992.

[8] R.J. Douglas, K.A.C. Martin, and D. Whitteridge. An intracellular analysis of the visual responses of neurones in cat visual cortex. *J. Physiology*, 440:659–696, 1991.

[9] D.J. Felleman and D.C. Van Essen. Distributed hierarchical processing in the primate cerebral cortex. *Cerebral Cortex*, 1:1–47, 1991.

[10] H.G. Kim and B.W. Connors. Apical dendrites of the neocortex: Correlation between sodium- and calcium-dependent spiking and pyramidal cell morphology. *J. Neuroscience*, In press.

[11] G. Laurent. A dendritic gain-control mechanism in axonless neurons of the locust, schistocerca americana. *J Physiology (London)*, 470:45–54, 1993.

[12] A. Mason and A.U. Larkman. Correlations between morphology and electrophysiology of pyramidal neurons in slices of rat visual cortex. II. Electrophysiology. *J. Neuroscience*, 10(5):1415–1428, 1990.

[13] H. Pockberger. Electrophysiological and morphological properties of rat motor cortex neurons in vivo. *Brain Research*, 539:181–190, 1991.

[14] P.K. Powers, R.F. Tobinson, and M.A. Konodi. Effective synaptic current can be estimated from measurements of neuronal discharge. *J. Neurophysiology*, 68(3):964–968, 1992.

[15] W.G. Regehr, J. Kehoe, P. Ascher, and C.M. Armstrong. Synaptically triggered action-potentials in dendrites. *Neuron*, 11(1):145–151, 1993.

[16] K.S. Rockland and A. Virga. Terminal arbors of individual "feedback" axons projecting from area V2 to V1 in the macaque monkey: a study using immunohistochemistry of anterogradely transported phaseoulus vulgaris-leucoagglutinin. *J. Comp. Neurol.*, 285:54–72, 1989.

[17] W.A. Spencer and E.R. Kandel. Electrophysiology of hippocampal neurons. IV fast prepotentials. *J. Neurophysiology*, 24:272–285, 1961.

[18] D.W. Tank, M. Sugimori, J.A. Connor, and R.R. Llinás. Spatially resolved calcium dynamics of mammalian purkinje cells in cerebellar slice. *Science*, 242:773–777, 1988.

[19] R. Yuste, K.R. Delaney, M.J. Gutnick, and D.W. Tank. Spatially localized calcium accumulations in apical dendrites of layer 5 neocortical neurons. In *Neuroscience Abstr. 19*, page 616.2, 1993.
